# An Information Maximization Model of Eye Movements

**Laura Walker Renninger, James Coughlan, Preeti Verghese**
Smith-Kettlewell Eye Research Institute
*{laura, coughlan, preeti}@ski.org*

**Jitendra Malik**
University of California, Berkeley
*malik@eecs.berkeley.edu*

## Abstract

We propose a sequential information maximization model as a general strategy for programming eye movements. The model reconstructs high-resolution visual information from a sequence of fixations, taking into account the fall-off in resolution from the fovea to the periphery. From this framework we get a simple rule for predicting fixation sequences: after each fixation, fixate next at the location that minimizes uncertainty (maximizes information) about the stimulus. By comparing our model performance to human eye movement data and to predictions from a saliency and random model, we demonstrate that our model is best at predicting fixation locations. Modeling additional biological constraints will improve the prediction of fixation sequences. Our results suggest that information maximization is a useful principle for programming eye movements.

## 1 Introduction

Since the earliest recordings [1, 2], vision researchers have sought to understand the non-random yet idiosyncratic behavior of volitional eye movements. To do so, we must not only unravel the bottom-up visual processing involved in selecting a fixation location, but we must also disentangle the effects of top-down cognitive factors such as task and prior knowledge. Our ability to predict volitional eye movements provides a clear measure of our understanding of biological vision.

One approach to predicting fixation locations is to propose that the eyes move to points that are "salient". Salient regions can be found by looking for center-surround contrast in visual channels such as color, contrast and orientation, among others [3, 4]. Saliency has been shown to correlate with human fixation locations when observers "look around" an image [5, 6] but it is not clear if saliency alone can explain why some locations are chosen over others and in what order. Task as well as scene or object knowledge will play a role in constraining the fixation

locations chosen [7]. Observations such as this led to the scanpath theory, which proposed that eye movement sequences are tightly linked to both the encoding and retrieval of specific object memories [8].

## 1.1   Our Approach

We propose that during natural, active vision, we center our fixation on the most *informative* points in an image in order to reduce our overall uncertainty about what we are looking at. This approach is intuitive and may be biologically plausible, as outlined by Lee & Yu [9]. The most informative point will depend on both the observer's current knowledge of the stimulus and the task. The quality of the information gathered with each fixation will depend greatly on human visual resolution limits. This is the reason we must move our eyes in the first place, yet it is often ignored. A sequence of eye movements may then be understood within a framework of *sequential information maximization*.

## 2   Human eye movements

We investigated how observers examine a novel shape when they must rely heavily on bottom-up stimulus information. Because eye movements will be affected by the task of the observer, we constructed a learn-discriminate paradigm. Observers are asked to carefully study a shape and then discriminate it from a highly similar one.

## 2.1   Stimuli and Design

We use novel silhouettes to reduce the influence of object familiarity on the pattern of eye movements and to facilitate our computations of information in the model. Each silhouette subtends 12.5º to ensure that its entire shape cannot be characterized with a single fixation.

During the learning phase, subjects first fixated a marker and then pressed a button to cue the appearance of the shape which appeared 10º to the left or right of fixation. Subjects maintained fixation for 300ms, allowing for a peripheral preview of the object. When the fixation marker disappeared, subjects were allowed to study the object for 1.2 seconds while their eye movements were recorded. During the discrimination phase, subjects were asked to select the shape they had just studied from a highly similar shape pair (Figure 1). Performance was near 75% correct, indicating that the task was challenging yet feasible. Subjects saw 140 shapes and given auditory feedback.

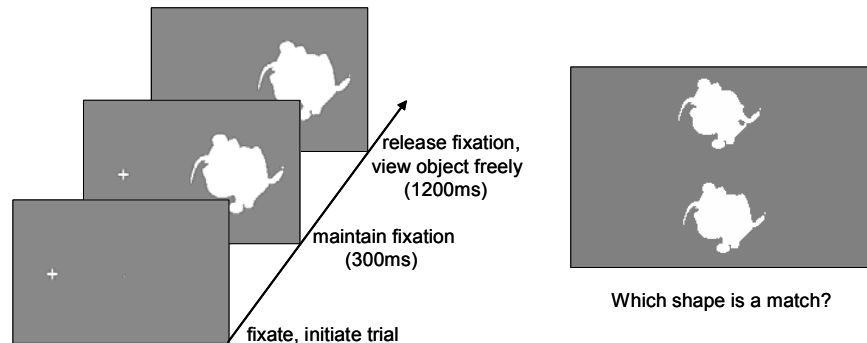

Figure 1. Temporal layout of a trial during the learning phase (left). Discrimination of learned shape from a highly similar one (right).

## 2.2 Apparatus

Right eye position was measured with an SRI Dual Purkinje Image eye tracker while subjects viewed the stimulus binocularly. Head position was fixed with a bitebar. A 25 dot grid that covered the extent of the presentation field was used for calibration. The points were measured one at a time with each dot being displayed for 500ms. The stimuli were presented using the Psychtoolbox software [10].

## 3  Model

We wish to create a model that builds a representation of a shape silhouette given imperfect visual information, and which updates its representation as new visual information is acquired. The model will be defined statistically so as to explicitly encode uncertainty about the current knowledge of the shape silhouette. We will use this model to generate a simple rule for predicting fixation sequences: after each fixation, fixate next at the location that will decrease the model's uncertainty as much as possible. Similar approaches have been described in an ideal observer model for reading [11], an information maximization algorithm for tracking contours in cluttered images [12] and predicting fixation locations during object learning [13].

### 3.1  Representing information

The information in silhouettes clearly resides at its contour, which we represent with a collection of points and associated tangent orientations. These points and their associated orientations are called *edgelets*, denoted $e_1$, $e_2$, ... $e_N$, where N is the total number of edgelets along the boundary. Each edgelet $e_i$ is defined as a triple $e_i=(x_i, y_i, z_i)$ where $(x_i, y_i)$ is the 2D location of the edgelet and $z_i$ is the orientation of the tangent to the boundary contour at that point. $z_i$ can assume any of Q possible values 1, 2, …, Q, representing a discretization of Q possible orientations ranging from 0 to $\pi$, and we have chosen Q=8 in our experiments. The goal of the model is to infer the most likely orientation values given the visual information provided by one or more fixations.

### 3.2  Updating knowledge

The visual information is based on indirect measurements of the true edgelet values $e_1$, $e_2$, ... $e_N$. Although our model assumes complete knowledge of the number N and locations $(x_i, y_i)$ of the edgelets, it does not have direct access to the orientations $z_i$.[1] Orientation information is instead derived from measurements that summarize the local frequency of occurrence of edgelet orientations, averaged locally over a coarse scale (corresponding to the spatial scale at which resolution is limited by the human visual system). These coarse measurements provide *indirect* information about individual edgelet orientations, which may not uniquely determine the orientations. We will use a simple statistical model to estimate the distribution of individual orientation values conditioned on this information.

Our measurements are defined to model the resolution limitations of the human visual system, with highest resolution at the fovea and lower resolution in the

periphery. Distance to the fovea is measured as eccentricity E, the visual angle between any point and the fovea. If $\vec{x} = (x, y)$ is the location of a point in an image and $\vec{f} = (f_x, f_y)$ is the fixation (i.e. foveal) location in the image then the eccentricity is $E = \|\vec{x} - \vec{f}\|$, measured in units of visual degrees. The effective resolution of orientation discrimination falls with increasing eccentricity as $r(E) = F_{PH}(E + E_2)$ where r(E) is an effective radius over which the visual system spatially pools information and $F_{PH}$ =0.1 and $E_2$=0.8 [14].

Our model represents pooled information as a histogram of edge orientations within the effective radius. For each edgelet $e_i$ we define the histogram of all edgelet orientations $e_j$ within radius $r_i = r(E)$ of $e_i$, where E is the eccentricity of $\vec{x}_i = (x_i, y_i)$ relative to the current fixation $\vec{f}$, i.e. $E = \|\vec{x}_i - \vec{f}\|$. To define the histogram more precisely we will introduce the neighborhood set $N_i$ of all indices j corresponding to edgelets within radius $r_i$ of $e_i$ : $N_i = \{all\ j\ s.t. \|\vec{x}_i - \vec{x}_j\| \le r_i\}$, with number of neighborhood edgelets $|N_i|$. The (normalized) histogram centered at edgelet $e_i$ is then defined as

$$h_{iz} = \frac{1}{|N_i|} \sum_{j \in N_i} \delta_{z, z_j} \,,$$

which is the proportion of edgelet orientations that assume value z in the (eccentricity-dependent) neighborhood of edgelet $e_i$.[2]

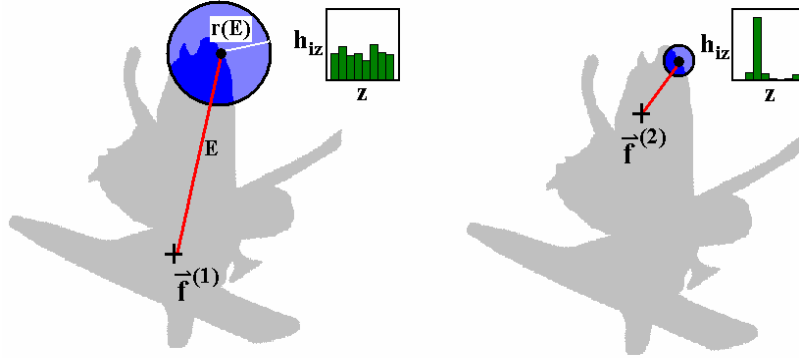

Figure 2. Relation between eccentricity E and radius r(E) of the neighborhood (disk) which defines the local orientation histogram ($h_{iz}$). Left and right panels show two fixations for the same object.

Up to this point we have restricted ourselves to the case of a single fixation. To designate a sequence of multiple fixations we will index them by k=1, 2, …, K (for K total fixations). The $k^{th}$ fixation location is denoted by $\vec{f}^{(k)} = (f_x^k, f_y^k)$. The quantities $r_i$, $N_i$ and $h_{iz}$ depend on fixation location and so to make this dependence explicit we will augment them with superscripts as $r_i^{(k)}$, $N_i^{(k)}$, and $h_{iz}^{(k)}$.

Now we describe the statistical model of edgelet orientations given information obtained from multiple fixations. Ideally we would like to model the exact distribution of orientations conditioned on the histogram data: $P(z_i, z_2, \ldots z_N \mid \{h_{iz}^{(1)}\}, \{h_{iz}^{(2)}\}, \ldots, \{h_{iz}^{(K)}\})$, where $\{h_{iz}^{(k)}\}$ represents all histogram components z at every edgelet $e_i$ for fixation $\vec{f}^{(k)}$. This exact distribution is intractable, so we will use a simple approximation. We assume the distribution factors over individual edgelets:

$$P(z_i, z_2, \ldots z_N \mid \{h_{iz}^{(1)}\}, \{h_{iz}^{(2)}\}, \ldots, \{h_{iz}^{(K)}\}) = \prod_{i=1}^{N} g_i(z_i)$$

where $g_i(z_i)$ is the marginal distribution of orientation $z_i$. Determining these marginal distributions is still difficult even with the factorization assumption, so we will make an additional approximation: $g_i(z_i) = \dfrac{1}{Z_i} \prod_{k=1}^{K} h_{iz}^{(k)}$, where $Z_i$ is a suitable normalization factor. This approximation corresponds to treating $h_{iz}^{(k)}$ as a likelihood function over z, with independent likelihoods for each fixation k. While the approximation has some undesirable properties (such as making the marginal distribution $g_i(z_i)$ more peaked if the same fixation is made repeatedly), it provides a simple mechanism for combining histogram evidence from multiple, distinct fixations.

## 3.3    Selecting the next fixation

Given the past K fixations, the next fixation $\vec{f}^{(K+1)}$ is chosen to minimize the model entropy of the edgelet orientations. In other words, $\vec{f}^{(K+1)}$ is chosen to minimize

$H(\vec{f}^{(K+1)}) = entropy[P(z_i, z_2, \ldots z_N \mid \{h_{iz}^{(1)}\}, \{h_{iz}^{(2)}\}, \ldots, \{h_{iz}^{(K+1)}\})]$, where the entropy of a distribution P(x) is defined as $-\sum_{x} P(x) \log P(x)$. In practice, we minimize the entropy by evaluating it across a set of candidate locations $\vec{f}^{(K+1)}$ which forms a regularly sampled grid across the image.[3] We note that this selection rule makes decisions that depend, in general, on the full history of previous K fixations.

## 4   Results

Figure 3 shows an example of one observer's eye movements superimposed over the shape (top row), the prediction from a saliency model (middle row) [3] and the prediction from the information maximization model (bottom row). The information maximization model updates its prediction after each fixation.

An ideal sequence of fixations can be generated by both models. The saliency model selects fixations in order of decreasing salience. The information maximization model selects the maximally informative point after incorporating information from the previous fixations. To provide an additional benchmark, we also implemented a

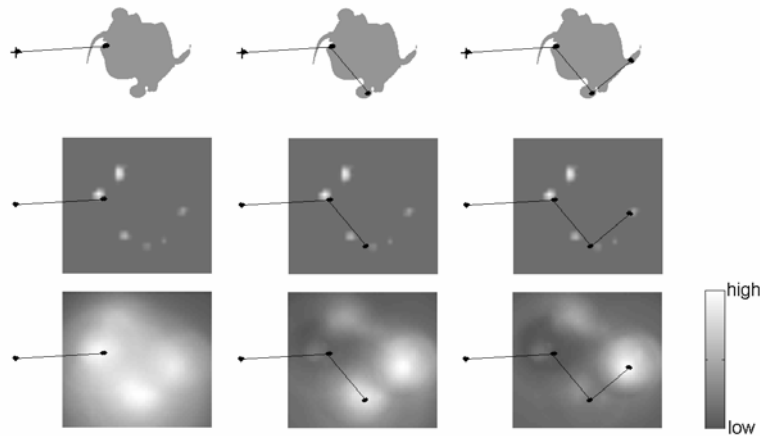

Figure 3. Example eye movement pattern, superimposed over the stimulus (top row), saliency map (middle row) and information maximization map (bottom row).

model that selects fixations at random. One way to quantify the performance is to map a subject's fixations onto the closest model predicted fixation locations, ignoring the sequence in which they were made. In this analysis, both the saliency and information maximization models are significantly better than random at predicting candidate locations ($p < 0.05$; t-test) for three observers (Figure 4, left). The information maximization model performs slightly but significantly better than the saliency model for two observers (lm, kr). If we match fixation locations while retaining the sequence, errors become quite large, indicating that the models cannot account for the observed behavior (Figure 4, right).

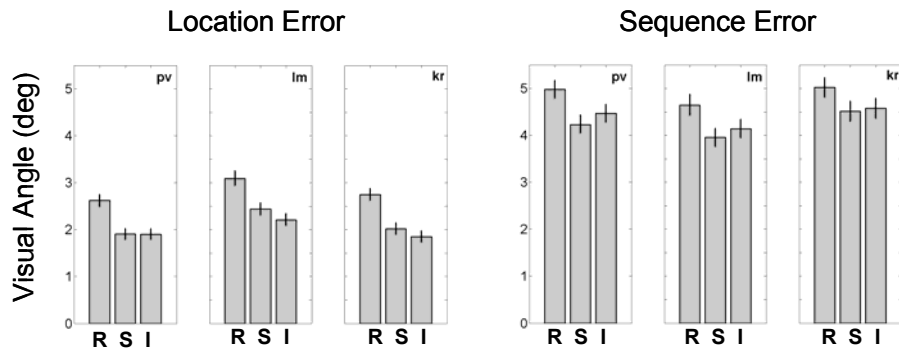

Figure 4. Prediction error of three models: random (R), saliency (S) and information maximization (I) for three observers (pv, lm, kr). The left panel shows the error in predicting fixation locations, ignoring sequence. The right panel shows the error when sequence is retained before mapping. Error bars are 95% confidence intervals.

The information maximization model incorporates resolution limitations, but there are further biological constraints that must be considered if we are to build a model that can fully explain human eye movement patterns. First, saccade amplitudes are typically around 2-4° and rarely exceed 15° [15]. When we move our eyes, the image of the visual world is smeared across the retina and our perception of it is actively suppressed [16]. Shorter saccade lengths may be a mechanism to reduce this cost. This biological constraint would cause a fixation to fall short of the prediction if it is distant from the current fixation (Figure 5).

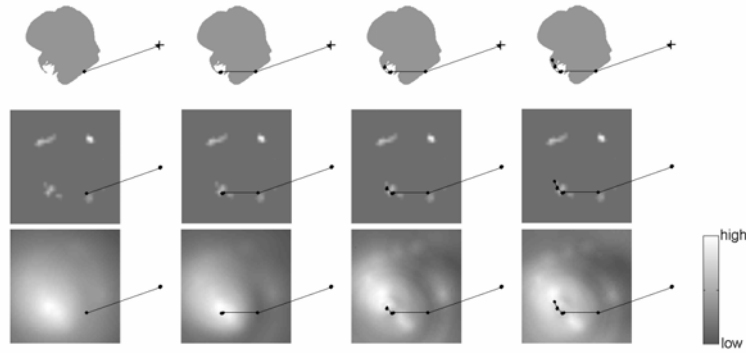

Figure 5. Cost of moving the eyes. Successive fixations may fall short of the maximally salient or informative point if it is very distant from the current fixation.

Second, the biological system may increase its sampling efficiency by planning a series of saccades concurrently [17, 18]. Several fixations may therefore be made before sampled information begins to influence target selection. The information maximization model currently updates after each fixation. This would create a discrepancy in the prediction of the eye movement sequence (Figure 6).

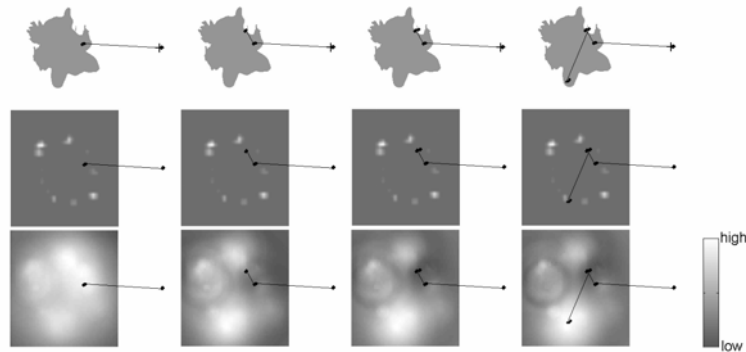

Figure 6. Three fixations are made to a location that is initially highly informative according to the information maximization model. By the fourth fixation, the subject finally moves to the next most informative point.

## 5  Discussion

Our model and the saliency model are using the same image information to determine fixation locations, thus it is not surprising that they are roughly similar in their performance of predicting human fixation locations. The main difference is how we decide to "shift attention" or program the sequence of eye movements to these locations. The saliency model uses a winner-take-all and inhibition-of-return mechanism to shift among the salient regions. We take a completely different approach by saying that observers adopt a strategy of sequential information maximization. In effect, the history of where we have been matters because our model is continually collecting information from the stimulus. *We have an implicit "inhibition-of-return" because there is little to be gained by revisiting a point.* Second, we attempt to take biological resolution limits into account when determining the quality of information gained with each fixation. By including additional biological constraints such as the cost of making large saccades and the

natural time course of information update, we may be able to improve our prediction of eye movement sequences.

We have shown that the programming of eye movements can be understood within a framework of sequential information maximization. This framework is portable to any image or task. A remaining challenge is to understand how different tasks constrain the representation of information and to what degree observers are able to utilize the information.

## Acknowledgments

Smith-Kettlewell Eye Research Institute, NIH Ruth L. Kirschstein NRSA, ONR #N00014-01-1-0890, NSF #IIS0415310, NIDRR #H133G030080, NASA #NAG 9-1461.

## Footnotes

[1] Although the visual system does not have precise knowledge of location coordinates, the model is greatly simplified by assuming this knowledge. It is reasonable to expect that location uncertainty will be highly correlated with orientation uncertainty, so that the inclusion of location should not greatly affect the model's decisions of where to fixate next.

[2] $\delta_{x,y}$ is the Kronecker delta function, defined to equal 1 if $x = y$ and 0 if $x \ne y$.

[3] This rule evaluates the entropy resulting from every possible next fixation before making a decision. Although this rule is suitable for our modeling purposes, it would be inefficient to implement in a biological or machine vision system. A practical decision rule would use current knowledge to estimate the *expected* (rather than actual) entropy.

## References

[1] Buswell (1935). *How people look at pictures*. Chicago: The University of Chicago Press.

[2] Yarbus (1967). *Eye movements and vision*. New York: Plenum Press.

[3] Itti & Koch (2000). A saliency-based search mechanism for overt and covert shifts of visual attention. *Vision Research, 40,* 1489-1506.

[4] Kadir & Brady (2001). Scale, saliency and image description. *International Journal of Computer Vision, 45(2),* 83-105.

[5] Parkhurst, Law, and Niebur (2002). Modeling the role of salience in the allocation of overt visual attention. *Vision Research, 42(1),* 107-123.

[6] Nothdurft (2002). Attention shifts to salient targets. *Vision Research, 42,* 1287-1306.

[7] Oliva, Torralba, Castelhano & Henderson (2003). Top-down control of visual attention in object detection. *Proceedings of the IEEE International Conference on Image Processing,* Barcelona, Spain.

[8] Noton & Stark (1971). Scanpaths in eye movements during pattern perception. *Science, 171,* 308-311.

[9] Lee & Yu (2000). An information-theoretic framework for understanding saccadic behaviors. *Advanced in Neural Processing Systems, 12,* 834-840.

[10] Brainard (1997). The psychophysics toolbox. *Spatial Vision, 10 (4),* 433-436.

[11] Legge, Hooven, Klitz, Mansfield & Tjan (2002). Mr.Chips 2002: new insights from an ideal-observer model of reading. *Vision Research, 42,* 2219-2234.

[12] Geman & Jedynak (1996). An active testing model for tracking roads in satellite images. *IEEE Trans. Pattern Analysis and Machine Intel, 18(1),* 1-14.

[13] Renninger & Malik (2004). Sequential information maximization can explain eye movements in an object learning task. *Journal of Vision, 4(8),* 744a.

[14] Levi, Klein & Aitesbaomo (1985). Vernier acuity, crowding and cortical magnification. *Vision Research, 25(7),* 963-977.

[15] Bahill, Adler & Stark (1975). Most naturally occurring human saccades have magnitudes of 15 degrees or less. *Investigative Ophthalmology, 14,* 468-469.

[16] Burr, Morrone & Ross (1994). Selective suppression of the magnocellular visual pathway during saccadic eye movements. *Nature, 371,* 511-513.

[17] Caspi, Beutter & Eckstein (2004). The time course of visual information accrual guiding eye movement decisions. *Proceedings of the Nat'l Academy of Science, 101(35),* 13086-90.

[18] McPeek, Skavenski & Nakayama (2000). Concurrent processing of saccades in visual search. *Vision Research, 40,* 2499-2516.
